# Efficient Methods for Privacy Preserving Face Detection

**Shai Avidan**
Mitsubishi Electric Research Labs
201 Broadway
Cambridge, MA 02139
avidan@merl.com

**Moshe Butman**
Department of Computer Science
Bar Ilan University
Ramat-Gan, Israel
butmanm@cs.biu.edu

## Abstract

Bob offers a face-detection web service where clients can submit their images for analysis. Alice would very much like to use the service, but is reluctant to reveal the content of her images to Bob. Bob, for his part, is reluctant to release his face detector, as he spent a lot of time, energy and money constructing it. Secure Multi-Party computations use cryptographic tools to solve this problem without leaking any information. Unfortunately, these methods are slow to compute and we introduce a couple of machine learning techniques that allow the parties to solve the problem while leaking a controlled amount of information. The first method is an information-bottleneck variant of AdaBoost that lets Bob find a subset of features that are enough for classifying an image patch, but not enough to actually reconstruct it. The second machine learning technique is active learning that allows Alice to construct an online classifier, based on a small number of calls to Bob's face detector. She can then use her online classifier as a fast rejector before using a cryptographically secure classifier on the remaining image patches.

## 1   Introduction

The Internet triggered many opportunities for cooperative computing in which buyers and sellers can meet to buy and sell goods, information or knowledge. Placing classifiers on the Internet allows buyers to enjoy the power of a classifier without having to train it themselves. This benefit is hindered by the fact that the seller, that owns the classifier, learns a great deal about the buyers' data, needs or goals. This raised the need for privacy in Internet transactions. While it is now common to assume that the buyer and the seller secure their data exchange from the rest of the world, we are interested in a stronger level of security that allows the buyer to hide his data from the seller as well. Of course, the same can be said about the seller, who would like to maintain the privacy of his hard-earned classifier.

Secure Multi-Party Computation (SMC) are based on cryptographic tools that let two parties, Alice and Bob, to engage in a protocol that will allow them to achieve a common goal, without revealing the content of their input. For example, Alice might be interested in classifying her data using Bobs' classifier without revealing anything to Bob, not even the classification result, and without learning anything about Bobs' classifier, other than a binary answer to her query.

Recently, Avidan & Butman introduced *Blind Vision* [1] which is a method for securely evaluating a Viola-Jones type face detector [12]. Blind Vision uses standard cryptographic tools and is painfully slow to compute, taking a couple of hours to scan a single image. The purpose of this work is to explore machine learning techniques that can speed up the process, at the cost of a controlled leakage of information.

In our hypothetical scenario Bob has a face-detection web service where clients can submit their images to be analyzed. Alice would very much like to use the service, but is reluctant to reveal the content of the images to Bob. Bob, for his part, is reluctant to release his face detector, as he spent a lot of time, energy and money constructing it.

In our face detection protocol Alice raster scans the image and sends every image patch to Bob to be classified. We would like to replace cryptographically-based SMC methods with Machine Learning algorithms that might leak some information but are much faster to execute. The challenge is to design protocols that can explicitly control the amount of information leaked. To this end we propose two, well known, machine learning techniques. One based on the information bottleneck and the other on active learning.

The first method is a privacy-preserving feature selection which is a variant of the information-bottleneck principle to find features that are useful for classification but not for signal reconstruction. In this case, Bob can use his training data to construct different classifiers that offer different trade-offs of information leakage versus classification accuracy. Alice can then choose the trade-off that suits her best and send only those features to Bob for classification. This method can be used, for example, as a filtering step that rejects a large number of the image patches as having no face included in them, followed by a SMC method that will securely classify the remaining image patches, using the full classifier that is known only to Bob.

The second method is active learning and it helps Alice choose which image patches to send to Bob for classification. This method can be used either with the previous method or directly with an SMC protocol. The idea being that instead of sending all image patches to Bob for classification, Alice might try to learn from the interaction as much as she can and use her online trained classifier to reject some of the image patches herself. This can minimize the amount of information revealed to Bob, if the parties use the privacy-preserving features or the computational load, if the parties are using cryptographically-based SMC methods.

## 2 Background

Secure multi-party computation originated from the work of Yao [14] who gave a solution to the millionaire problem: Two parties want to find which one has a larger number, without revealing anything else about the numbers themselves. Later, Goldriech *et al.* [5] showed that any function can be computed in such a secure manner. However, the theoretical construct was still too demanding to be of practical use. An easy introduction to Cryptography is given in [9] and a more advanced and theoretical treatment is given in [4]. Since then many secure protocols were reported for various data mining applications [7, 13, 1]. A common assumption in SMC is that the parties are *honest but curious*, meaning that they will follow the agreed-upon protocol but will try to learn as much as possible from the data-flow between the two parties. We will follow this assumption here.

The information bottleneck principle [10] shows how to compress a signal while preserving its information with respect to a target signal. We offer a variant of the self-consistent equations used to solve this problem and offer a greedy feature selection algorithm that satisfy privacy constraints, that are represented as the percentage of the power spectrum of the original signal.

Active learning methods assume that the student (Alice, in our case) has access to an Oracle (Bob) for labeling. The usual motivation in active learning is that the Oracle is assumed to be a human operator and having him label data is a time consuming task that should be avoided. Our motivation is similar, Alice would like to avoid using Bob because of the high computational cost involved in case of cryptographically secure protocols, or for fear of leaking information in case non-cryptographic methods are used. Typical active learning applications assume that the distribution of class size is similar [2, 11]. A notable exception is the work of [8] that propose an active learning method for anomaly detection. Our case is similar as image patches that contain faces are rare in an image.

## 3 Privacy-preserving Feature Selection

Feature selection aims at finding a subset of the features that optimize some objective function, typically a classification task [6]. However, feature selection does not concern itself with the correlation of the feature subset with the original signal.

This is handled with the information bottleneck method [10], that takes a joint distribution $p(x, y)$ and finds a compressed representation of $X$, denoted by $T$, that is as informative about $Y$. This is achieved by minimizing the following functional:

$$\min_{p(t|x)} \mathcal{L} : \mathcal{L} \equiv I(X;T) - \beta I(T;Y) \tag{1}$$

where $\beta$ is a trade-off parameter that controls the trade off between compressing $X$ and maintaining information about $Y$. The functional $\mathcal{L}$ admits a set of self-consistent equations that allows one to find a suitable solution.

We map the information bottleneck idea to a feature selection algorithm to obtain a Privacy-preserving Feature Selection (PPFS) and describe how Bob can construct such a feature set. Let Bob have a training set of image patches, their associated label and a *weight* associated with every feature (pixel) denoted $\{\mathbf{x_n}, y_n, s_n\}_{n=1}^{N}$. Bob's goal is to find a feature subset $\mathcal{I} \equiv \{i_1, \ldots, i_k\}$ s.t. a classifier $F(\mathbf{x}(\mathcal{I}))$ will minimize the classification error, where $\mathbf{x}(\mathcal{I})$ denotes a sample $\mathbf{x}$ that uses only the features in the set $\mathcal{I}$. Formally, Bob needs to minimize:

$$\min_{F} \sum_{n=1}^{N} (F(\mathbf{x_n}(\mathcal{I})) - y_n))^2 \tag{2}$$

$$\text{subject to} \sum_{i \in \mathcal{I}} s_i < \Lambda$$

where $\Lambda$ is a user defined threshold that defines the amount of information that can be leaked.

We found it useful to use the PCA spectrum to measure the amount of information. Specifically, Bob computes the PCA space of all the face images in his database and maps all the data to that space, without reducing dimensionality. The weights $\{s_n\}_{n=1}^{N}$ are now set to the eigenvalues associated with each dimension in the PCA space. This avoids the need to compute the mutual information between pixels by making the assumption that features do not carry mutual information with other features beyond second order statistics.

---

**Algorithm 1** Privacy-Preserving Feature Selection

---

Input: $\{\mathbf{x_n}, y_n, s_n\}_{n=1}^{N}$
        Threshold $\Lambda$
        Number of iterations $T$
Output:   A privacy-preserving strong classifier $F(\mathbf{x})$

- Start with weights $w_n = 1/N \quad n = 1, 2, \ldots, N$, $F(\mathbf{x}) = 0, \mathcal{I} = \emptyset$
- Repeat for $t = 1, 2, \ldots, T$
    - Set working index set $\mathcal{J} = \mathcal{I} \cup \{j | s_j + \sum_{i \in \mathcal{I}} s_i < \Lambda\}$
    - Repeat for $j \in \mathcal{J}$
        * Fit a regression stump $g_j(\mathbf{x}(j)) \equiv a_j(\mathbf{x}(j) > \theta_j) + b_j$ to the $j$-th feature, $\mathbf{x}(j)$
        * Compute error $e_j = \frac{\sum_{n=1}^{N} w_n(y_n - (a_j(\mathbf{x_n}(j) > \theta_j + b_j)^2}{\sum_{n=1}^{N} w_n}$
    - Set $f_t = g_i$ where $e_i < e_j \quad \forall j \in \mathcal{J}$
    - Update:

$$F(\mathbf{x}) \leftarrow F(\mathbf{x}) + f_t(\mathbf{x}) \tag{3}$$

$$w_n \leftarrow w_n e^{-y_n f_t(\mathbf{x_n})} \tag{4}$$

$$\mathcal{I} \leftarrow \mathcal{I} \cup \{i\} \tag{5}$$

---

Boosting was used for feature selection before [12] and Bob takes a similar approach here. He uses a variant of the gentleBoost algorithm [3] to find a greedy solution to (2). Specifically, Bob uses gentleBoost with "stumps" as the weak classifiers where each "stump" works on only one feature. The only difference from gentleBoost is in the choice of the features to be selected. In the original algorithm all the features are evaluated in every iteration of the algorithm, but here Bob can only use

a subset of the features. In each iteration Bob can use features that were already selected or those that adding them will not increase the total weight of selected features beyond the threshold $\Lambda$.

Once Bob computed the privacy-preserving feature subset, the amount of information it leaks and its classification accuracy he publishes this information on the web. Alice then needs to map her image patches to this low-dimensional privacy-preserving feature space and send the data to Bob for classification.

# 4   Privacy-Preserving Active Learning

In our face detection example Alice needs to submit many image patches to Bob for classification. This is computationally expensive if SMC methods are used and reveals information, in case the privacy-preserving feature selection method discussed earlier is used. Hence, it would be beneficial if Alice could minimize the number of image patches she needs to send Bob for classification. This is where she might use active learning. Instead of raster scanning the image and submitting every image patch for classification she sends a small number of randomly selected image patches, and based on their label, she determines the next group of image patches to be sent for classification. We found that substantial gains can be made this way.

Specifically, Alice maintains an RBF network that is trained on-line, based on the list of labeled prototypes. Let $\{c_j, y_j\}_{j=1}^M$ be the list of $M$ prototypes that were labeled so far. Then, Alice constructs a kernel matrix $\mathbf{K}$ where $\mathbf{K_{ij}} = k(c_i, c_j)$ and solves the least squares equation $\mathbf{Ku} = \mathbf{y}$, where $\mathbf{y} = [y_1, \ldots, y_M]^T$. The kernel Alice uses is a Gaussian kernel whose width is set to be the range of the prototype coordinates, in each dimension. The score of each image patch $\mathbf{x}$ is given by $h(\mathbf{x}) = [k(\mathbf{x}, c_1), \ldots, k(\mathbf{x}, c_M)]\mathbf{u}$.

For the next round of classification Alice chooses the image patches with the highest $h(\mathbf{x})$ score. This is in line with [2, 11, 8] that consider choosing the examples of which one has the least amount of information. In our case, Alice is interested in finding image patches that contain faces (which we assume are labeled $+1$) but most of the prototypes will be labeled $-1$, because faces are a rear event in an image. As long as Alice does not sample a face image patch she will keep exploring the space of image patches in her image, by sampling image patches that are farthest away from the current set of prototypes. If an image patch that contains a face is sampled, then her online classifier $h(\mathbf{x})$ will label similar image patches with a high score, thus guiding the search towards other image patches that might contain a face. To avoid large overlap between patches, we force a minimum distance, in the image plane, between selected patches. The algorithm is given in algorithm 2.

---

**Algorithm 2** Privacy-Preserving Active Learning

---

Input: $\{\mathbf{x_i}\}_{i=1}^N$ unlabeled samples
        Number $M$ of classification calls allowed
        Number $s$ of samples to classify in each iteration
Output:   Online classifier $h(\mathbf{x})$

- Choose $s$ random samples $\{\mathbf{x_i}\}_{i=1}^s$, set $\mathbf{C} = [\mathbf{x_1}, \ldots, \mathbf{x_s}]$ and obtain their labels $\mathbf{y} = [y_1, \ldots, y_s]$ from Bob.
- Repeat for $m = 1, 2, ..., M$ times
  - Construct the kernel matrix $\mathbf{K_{ij}} = k(c_i, c_j)$ and solve for the weight vector $\mathbf{u}$ through least squares $\mathbf{Ku} = \mathbf{y}$.
  - Evaluate $h(\mathbf{x_i}) = [k(\mathbf{x_i}, c_1), \ldots, k(\mathbf{x_i}, c_m)]\mathbf{u} \quad \forall i = 1, \ldots, N$.
  - Choose top $s$ samples with highest $h(\mathbf{x})$ score, send them to Bob for classification and add them, and their labels to $\mathbf{C}, \mathbf{y}$, respectively.

---

# 5   Experiments

We have conducted a couple of experiments to validate both methods.

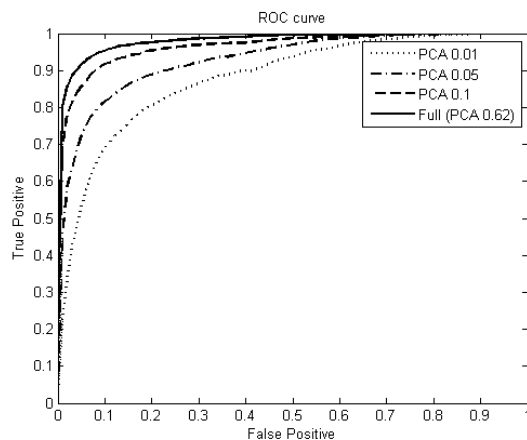

Figure 1: Privacy preserving feature selection. We show the ROC curves of four strong classifiers, each trained with 100 weak, "stump" classifiers, but with different levels of information leakage. The information leakage is defined as the amount of PCA spectrum captured by the features used in each classifier. The number in parenthesis shows how much of the eigen spectrum is captured by the features used in each classifier.

The first experiment evaluates the privacy-preserving feature selection method. The training set consisted of 9666 image patches of size $24 \times 24$ pixels each, split evenly to face/no-face images. The test set was of similar size. We then run algorithm 1 with different levels of the threshold $\Lambda$ and created a strong classifier with 100 weak, "stump" based, classifiers. The ROC curves of several such classifiers are shown in figure 1. We found that, for this particular dataset, setting $\Lambda = 0.1$ gives identical results to a full classifier, without any privacy constraints. Reducing $\Lambda$ to 0.01 did hurt the classification performance somewhat.

The second experiment tests the active learning approach. We assume that Alice and Bob use the classifier with $\Lambda = 0.05$ from the previous experiment, and measure how effective is the on-line classifier that Alice constructs in rejecting no-face image patches.

Recall that there are three classifiers at play. One is the full classifier that Bob owns, the second is the privacy-preserving classifier that Bob owns and the last is the on-line classifier that Alice constructs. Alice uses the labels of Bobs' privacy-preserving classifier to construct her on-line classifier and the questions is: how many image patches she can reject, without actually rejecting image patches that will be classified as faces by the full classifier (that she knows nothing about)?

Before we performed the experiment, we conducted the following pre-processing operation: We find, for each image, the scale at which the largest number of faces are detected using Bob's full classifier, and used only the image at that scale.

The experiment proceeds as follows. Alice chooses 5 image patches in each round, maps them to the reduced PCA space and sends them to Bob for classification, using his privacy-preserving classifier. Based on his labels, Alice then picks the next 5 image patches according to algorithm 2 and so on. Alice repeats the process 10 times, resulting in 50 patches that are sent to Bob for classification. The first 5 patches are chosen at random. Figure 2 shows the 50 patches selected by Alice, the online classifier $h$ and the corresponding rejection/recall curve, for several test images. The rejection/recall curve shows how many image patches Alice can safely reject, based on $h$, without rejecting a face that will be detected by Bobs' full classifier. For example, in the top row of figure 2 we see that rejecting the bottom $40\%$ of image patches based on the on-line classifier $h$ will not reject any face that can be detected with the full classifier. Thus 50 image patches that can be quickly labeled while leaking very little information can help Alice reject thousands of image patches.

Next, we conducted the same experiment on a larger set of images, consisting of 65 of the CMU+MIT database images[1]. Figure 3 shows the results. We found that, on average (dashed line),

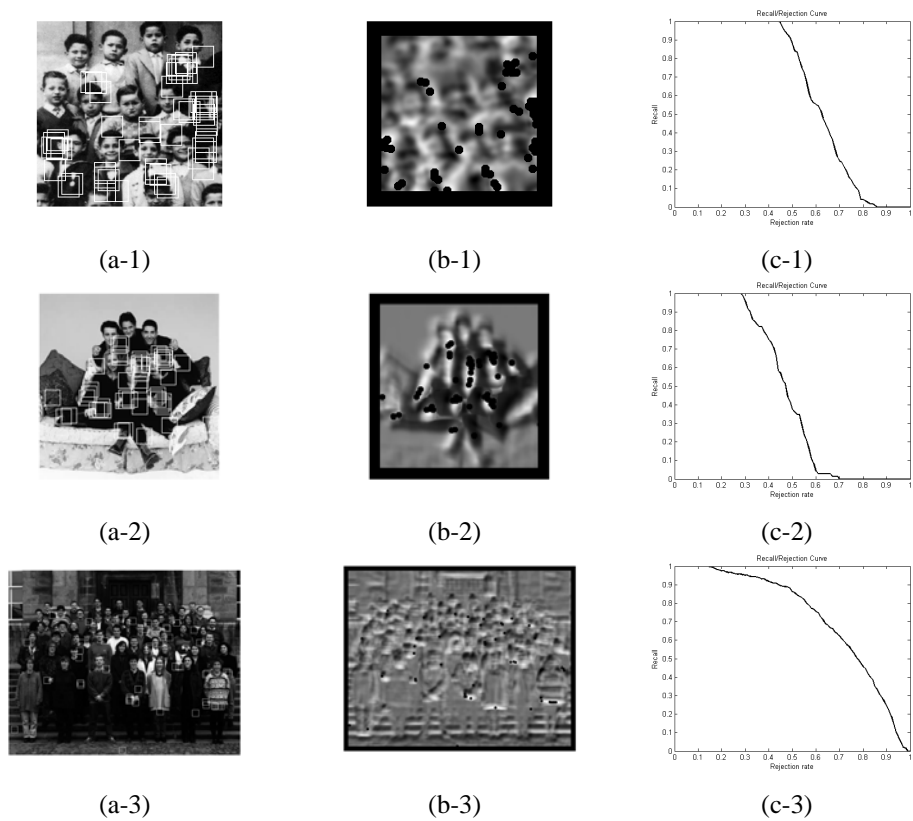

(a-1)    (b-1)    (c-1)

(a-2)    (b-2)    (c-2)

(a-3)    (b-3)    (c-3)

Figure 2: Examples of privacy-preserving feature selection and active learning. (a) The input images and the image patches (marked with white rectangles) selected by the active learning algorithm. (b) The response image computed by the online classifier (the black spots correspond to the position of the selected image patches). Brighter means a higher score. (c) The rejection/recall curve showing how many image patches can be safely rejected. For example, panel (c-1) shows that Alice can reject almost $50\%$ of the image patches, based on her online classifier (i.e., response image), and not miss a face that can be detected by the full classifier (that is known to Bob and not to Alice).

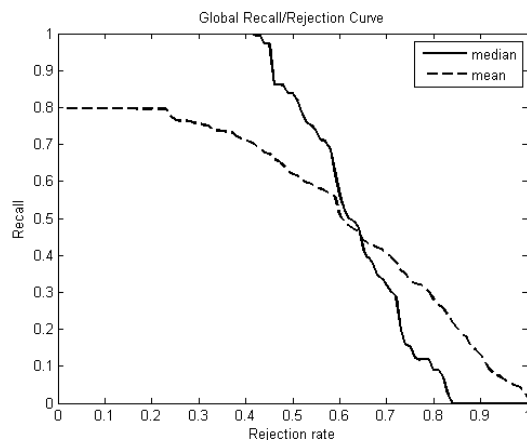

(a)

Figure 3: Privacy preserving active learning. Results on a dataset of 65 images. The figure shows how many image patches can be rejected, based on the online classifier that Alice owns, without rejecting a face. The horizontal axis shows how many image patches are rejected, based on the on-line classifier, and the vertical axis shows how many faces are maintained. For example, the figure shows (dashed line) that rejecting $20\%$ of all image patches, based on the on-line classifier, will maintain $80\%$ of all faces. The solid line shows that rejecting $40\%$ of all image patches, based on the on-line classifier, will not miss a face in at least half (i.e. the median) of the images in the dataset.

using only 50 labeled image patches Alice can reject up to about $20\%$ of the image patches in an image while keeping $80\%$ of the faces in that image (i.e., Alice will reject $20\%$ of the image patches that Bob's full classifier will classify as a face). If we look at the median (solid line), we see that for at least half the images in the data set, Alice can reject a little more than $40\%$ of the image patches without erroneously rejecting a face.

We found that increasing the number of labeled examples from 50 to a few hundreds does not greatly improve results, unless many thousands of samples are labeled, at which point too much information might be leaked.

## 6  Conclusions

We described two machine learning methods to accelerate cryptographically secure classification protocols. The methods greatly accelerate the performance of the system, while leaking a controlled amount of information. The two methods are a privacy preserving feature selection that is similar to the information bottleneck and an active learning technique that was found to be useful in learning a rejector from an extremely small number of labeled data. We plan to keep investigating these methods, apply them to classification tasks in other domains and develop new methods to make secure classification faster to use.

## Footnotes

[1]We used the 65 images in the newtest directory of the CMU+MIT dataset

## References

[1] S. Avidan and M. Butman. Blind vision. In *Proc. of European Conference on Computer Vision*, 2006.

[2] Y. Baram, R. El-Yaniv, and K. Luz. Online choice of active learning algorithms. *Journal of Machine Learning Research*, 5:255–291, March 2004.

[3] J. Friedman, T. Hastie, and R. Tibshirani. Additive logistic regression: a statistical view of boosting, 1998.

[4] O. Goldreich. *Foundations of Cryptography: Volume 1, Basic Tools*. Cambridge University Press, New York, 2001.

[5] O. Goldreich, S. Micali, and A. Wigderson. How to play any mental game or a completeness theorem for protocols with honest majority. In *ACM Symposium on Theory of Computing*, pages 218–229, 1987.

[6] I. Guyon and A. Elisseeff. An introduction to variable and feature selection. *Journal of Machine Learning Research*, 3:1157–1182, 2003.

[7] Y. Lindell and B. Pinkas. Privacy preserving data mining. In *CRYPTO: Proceedings of Crypto*, 2000.

[8] D. Pelleg and A. Moore. Active learning for anomaly and rare-category detection. In *In Advances in Neural Information Processing Systems 18*, 2004.

[9] B. Schneier. *Applied Cryptography*. John Wiley & Sons, New York, 1996.

[10] N. Tishby, F. Pereira, and W. Bialek. The information bottleneck method. In *In* Proc. of 37th Allerton Conference on communication and computation, 1999.

[11] S. Tong and D. Koller. Support vector machine active learning with applications to text classification. *Journal of Machine Learning Research*, 2:45–66, 2001.

[12] P. Viola and M. Jones. Rapid object detection using a boosted cascade of simple features. In *Coference on Computer Vision and Pattern Recognition (CVPR)*, 2001.

[13] R. N. Wright and Z. Yang. "privacy-preserving bayesian network structure computataion on distributed heterogeneous data". In *KDD '04: Proceeding of the tenth ACM SIGKDD international conference on Knowledge discovery in data mining*, pages 22–25, 2004.

[14] A. C. Yao. Protocols for secure computations. In *Proc. 23rd IEEE Symp. on Foundations of Comp. Science*, pages 160–164, Chicago, 1982. IEEE.
